# The Mondrian Process

**Daniel M. Roy**
Massachusetts Institute of Technology
droy@mit.edu

**Yee Whye Teh**
Gatsby Unit, University College London
ywteh@gatsby.ucl.ac.uk

## Abstract

We describe a novel class of distributions, called Mondrian processes, which can be interpreted as probability distributions over $k$d-tree data structures. Mondrian processes are multidimensional generalizations of Poisson processes and this connection allows us to construct multidimensional generalizations of the stick-breaking process described by Sethuraman (1994), recovering the Dirichlet process in one dimension. After introducing the Aldous-Hoover representation for jointly and separately exchangeable arrays, we show how the process can be used as a nonparametric prior distribution in Bayesian models of relational data.

## 1 Introduction

Relational data are observations of relationships between sets of objects and it is therefore natural to consider representing relations[1] as arrays of random variables, e.g., $(R_{i,j})$, where $i$ and $j$ index objects $x_i \in X$ and $y_j \in Y$. Nonrelational data sets (e.g., observations about individual objects in $X$) are simply one-dimensional arrays $(R_i)$ from this viewpoint.

A common Bayesian approach in the one-dimensional setting is to assume there is cluster structure and use a mixture model with a prior distribution over partitions of the objects in $X$. A similar approach for relational data would naïvely require a prior distribution on partitions of the product space $X \times Y = \{(x, y) \mid x \in X, y \in Y\}$. One choice is to treat each pair $(x, y)$ atomically, clustering the product space directly, e.g., by placing a Chinese restaurant process (CRP) prior on partitions of $X \times Y$. An unsatisfactory implication of this choice is that the distribution on partitions of $(R_{i,j})$ is exchangeable, i.e., invariant to swapping any two entries; this implies that the identity of objects is ignored when forming the partition, violating common sense.

*Stochastic block models*[2] place prior distributions on partitions of $X$ and $Y$ separately, which can be interpreted as inducing a distribution on partitions of the product space by considering the product of the partitions. By arranging the rows and columns of $(R_{i,j})$ so that clustered objects have adjacent indices, such partitions look like regular grids (Figure 1.1). An unfortunate side effect of this form of prior is that the "resolution" needed to model fine detail in one area of the array necessarily causes other parts of the array to be dissected, even if the data suggest there is no such structure. The annotated hierarchies described by Roy et al. (2007) generate random partitions which are not constrained to be regular grids (Figure 1.2), but the prior is inconsistent in light of missing data.

Motivated by the need for a consistent distribution on partitions of product spaces with more structure than classic block models, we define a class of nonparametric distributions we have named Mondrian processes after Piet Mondrian and his abstract grid-based paintings. Mondrian processes are random partitions on product spaces not constrained to be regular grids. Much like $k$d-trees, Mondrian processes partition a space with nested, axis-aligned cuts; see Figure 1.3 for examples.

We begin by introducing the notion of *partially exchangeable arrays* by Aldous (1981) and Hoover (1979), a generalization of exchangeability on sequences appropriate for modeling relational data.

We then define the Mondrian process, highlight a few of its elegant properties, and describe two nonparametric models for relational data that use the Mondrian process as a prior on partitions.

## 2 Exchangeable Relational Data

The notion of exchangeability[3], that the probability of a sequence of data items does not depend on the ordering of the items, has played a central role in hierarchical Bayesian modeling (Bernardo and Smith, 1994). A classic result by de Finetti (1931), later extended by Ryll-Nardzewski (1957), states that if $x_1, x_2, \ldots$ is an exchangeable sequence, then there exists a random parameter $\theta$ such that the sequence is conditionally iid given $\theta$:

$$p(x_1, \ldots, x_n) = \int p_\theta(\theta) \prod_{i=1}^n p_x(x_i|\theta)d\theta \tag{1}$$

That is, exchangeable sequences arise as a mixture of iid sequences, where the mixing distribution is $p(\theta)$. The notion of exchangeability has been generalized to a wide variety of settings. In this section we describe notions of exchangeability for relational data originally proposed by Aldous (1981) and Hoover (1979) in the context of exchangeable arrays. Kallenberg (2005) significantly expanded on the concept, and Diaconis and Janson (2007) showed a strong correspondence between such exchangeable relations and a notion of limits on graph structures (Lovász and Szegedy, 2006).

Here we shall only consider binary relations—those involving pairs of objects. Generalizations to relations with arbitrary arity can be gleaned from Kallenberg (2005). For $i, j = 1, 2, \ldots$ let $R_{i,j}$ denote a relation between two objects $x_i \in X$ and $y_j \in Y$ from possibly distinct sets $X$ and $Y$. We say that $R$ is *separately exchangeable* if its distribution is invariant to separate permutations on its rows and columns. That is, for each $n, m \geq 1$ and each pair of permutations $\pi \in S_n$ and $\sigma \in S_m$,

$$p(R_{1:n,1:m}) = p(R_{\pi(1:n),\sigma(1:m)}) \tag{2}$$

in MATLAB notation. Aldous (1981) and Hoover (1979) showed that separately exchangeable relations can always be represented in the following way: each object $i$ (and $j$) has a latent representation $\xi_i$ ($\eta_j$) drawn iid from some distribution $p_\xi$ ($p_\eta$); independently let $\theta$ be an additional random parameter. Then,

$$p(R_{1:n,1:m}) = \int p_\theta(\theta) \prod_i p_\xi(\xi_i) \prod_j p_\eta(\eta_j) \prod_{i,j} p_R(R_{i,j}|\theta, \xi_i, \eta_j)d\theta d\xi_{1:n}d\eta_{1:m} \tag{3}$$

As opposed to (1), the variables $\xi_i$ and $\eta_j$ capture additional dependencies specific to each row and column. If the two sets of objects are in fact the same, i.e. $X = Y$, then the relation $R$ is a square array. We say $R$ is *jointly exchangeable* if it is invariant to jointly permuting rows and columns; that is, for each $n \geq 1$ and each permutation $\pi \in S_n$ we have

$$p(R_{1:n,1:n}) = p(R_{\pi(1:n),\pi(1:n)}) \tag{4}$$

Such jointly exchangeable relations also have a form similar to (3). The differences are that we have one latent variable $\xi_i$ for to each object $x_i$, and that $R_{i,j}$, $R_{j,i}$ need not be independent anymore:

$$p(R_{1:n,1:n}) = \int p_\theta(\theta) \prod_i p_\xi(\xi_i) \prod_{i \leq j} p_R(R_{i,j}, R_{j,i}|\theta, \xi_i, \xi_j)d\theta d\xi_{1:n} \tag{5}$$

In (5) it is important that $p_R(s,t|\theta, \xi_i, \xi_j) = p_R(t,s|\theta, \xi_j, \xi_i)$ to ensure joint exchangeability. The first impression from (5) is that joint exchangeability implies a more restricted functional form than separately exchangeable (3). In fact, the reverse holds—(5) means that the latent representations of row $i$ and column $i$ need not be independent, and that $R_{i,j}$ and $R_{j,i}$ need not be conditionally independent given the row and column representations, while (3) assumes independence of both. For example, a *symmetric* relation, i.e. $R_{i,j} = R_{j,i}$, can only be represented using (5).

The above Aldous-Hoover representation serves as the theoretical foundation for hierarchical Bayesian modeling of exchangeable relational data, just as de Finetti's representation serves as a foundation for the modeling of exchangeable sequences. In Section 5, we cast the Infinite Relational Model (Kemp et al., 2006) and a model based on the Mondrian process into this representation.

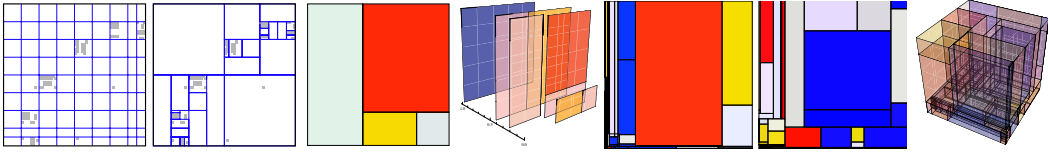

**Figure 1:** (1) Stochastic block models like the Infinite Relational model (Kemp et al., 2006) induce regular partitions on the product space, introducing structure where the data do not support it. (2) Axis-aligned partitions, like those produced by annotated hierarchies and the Mondrian process provide (a posteriori) resolution only where it is needed. (3) Mondrian process on unit square, $[0,1]^2$. (4) We can visualize the sequential hierarchical process by spreading the cuts out over time. The third dimension is $\lambda$. (5) Mondrian process with beta Lévy measure, $\mu(dx) = x^{-1}dx$ on $[0,1]^2$. (6) 10x zoom of 5 at origin. (7) Mondrian on $[\epsilon,1]^3$ with beta measure.

## 3 The Mondrian Process

The Mondrian process can be expressed as a recursive generative process that randomly makes axis-aligned cuts, partitioning the underlying product space in a hierarchical fashion akin to decision trees or $k$d-trees. The distinguishing feature of this recursive stochastic process is that it assigns probabilities to the various events in such a way that it is *consistent* (in a sense we make precise later). The implication of consistency is that we can extend the Mondrian process to infinite spaces and use it as a nonparametric prior for modeling exchangeable relational data.

### 3.1 The one dimensional case

The simplest space to introduce the Mondrian process is the unit interval $[0,1]$. Starting with an initial "budget" $\lambda$, we make a sequence of cuts, splitting the interval into subintervals. Each cut costs a random amount, eventually exhausting the budget and resulting in a finite partition $m$ of the unit interval. The cost, $E_I$, to cut an interval $I$ is exponentially distributed with inverse mean given by the length of the interval. Therefore, the first cut costs $E_{[0,1]} \sim \text{Exp}(1)$. Let $\lambda' = \lambda - E_{[0,1]}$. If $\lambda' < 0$, we make no cuts and the process returns the trivial partition $m = \{[0,1]\}$. Otherwise, we make a cut uniformly at random, splitting the unit interval into two subintervals $A$ and $B$. The process recurses independently on $A$ and $B$, with independent budgets $\lambda'$, producing partitions $m_A$ and $m_B$, which are then combined into a partition $m = m_A \bigcup m_B$ of $[0,1]$.

The resulting cuts can be shown to be a Poisson (point) process. Unlike the standard description of the Poisson process, the cuts in this "break and branch" process are organized in a hierarchy. As the Poisson process is a fundamental building block for random measures such as the Dirichlet process (DP), we will later exploit this relationship to build various multidimensional generalizations.

### 3.2 Generalizations to higher dimensions and trees

We begin in two dimensions by describing the generative process for a Mondrian process $m \sim \text{MP}(\lambda, (a,A), (b,B))$ on the rectangle $(a,A) \times (b,B)$. Again, let $\lambda' = \lambda - E$, where $E \sim \text{Exp}(A - a + B - b)$ is drawn from an exponential distribution with rate the sum of the interval lengths. If $\lambda' < 0$, the process halts, and returns the trivial partition $\{(a,A) \times (b,B)\}$. Otherwise, an axis-aligned cut is made uniformly at random along the combined lengths of $(a,A)$ and $(b,B)$; that is, the cut lies along a particular dimension with probability proportional to its length, and is drawn uniformly within that interval. W.l.o.g., a cut $x \in (a,A)$ splits the interval into $(a,x)$ and $(x,A)$. The process then recurses, generating independent Mondrian processes with diminished rate parameter $\lambda'$ on both sides of the cut: $m_< \sim \text{MP}(\lambda', (a,x), (b,B))$ and $m_> \sim \text{MP}(\lambda', (x,A), (b,B))$. The partition on $(a,A) \times (b,B)$ is then $m_< \bigcup m_>$. Like the one-dimensional special case, the $\lambda$ parameter controls the number of cuts, with the process more likely to cut rectangles with large perimeters.

The process can be generalized in several ways. In higher dimensions, the cost $E$ to make an additional cut is exponentially distributed with rate given by the sum over all dimensions of the interval lengths. Similarly, the cut point is chosen uniformly at random from all intervals, splitting only that interval in the recursion. Like non-homogeneous Poisson processes, the cut point need not

be chosen uniformly at random, but can instead be chosen according to a non-atomic rate measure $\mu_d$ associated with each dimension. In this case, lengths $(A - a)$ become measures $\mu_1(a, A)$.

The process can also be generalized beyond products of intervals. The key property of intervals that the Mondrian process relies upon is that any point cuts the space into one-dimensional, simply-connected pieces. Trees also have this property: a cut along an edge splits a tree into two trees. We denote a Mondrian process $m$ with rate $\lambda$ on a product of one-dimensional, simply-connected domains $\Theta_1 \times \cdots \times \Theta_D$ by $m \sim \mathrm{MP}(\lambda, \Theta_1, \ldots, \Theta_D)$, with the dependence on $\mu_1, \ldots, \mu_D$ left implicit. A description of the recursive generative model for the conditional Mondrian (see Section 4) is given in Algorithm 1.

## 4    Properties of the Mondrian Process

This section describes a number of interesting properties of the Mondrian process. The most important properties of the Mondrian is its self-consistency. Instead of representing a draw from a Mondrian as an unstructured partition of $\Theta_1 \times \cdots \times \Theta_D$, we will represent the whole history of the generative process. Thus a draw from the Mondrian process is either a trivial partition or a tuple $m = \langle d, x, \lambda', m_<, m_> \rangle$, representing a cut at $x$ along the $d$'th dimension $\Theta_d$, with nested Mondrians $m_<$ and $m_>$ on either side of the cut. Therefore, $m$ is itself a tree of axis-aligned cuts (a $k$d-tree data structure), with the leaves of the tree forming the partition of the original product space.

**Conditional Independencies**: The generative process for the Mondrian produces a tree of cuts, where each subtree is itself a draw from a Mondrian. The tree structure precisely reflects the conditional independencies of the Mondrian; e.g., the two subtrees $m_<$ and $m_>$ are conditional independent given $\lambda'$, $d$ and $x$ at the first cut.

**Consistency**: The Mondrian process satisfies an important self-consistency property: given a draw from a Mondrian on some domain, the partition on any subdomain has the same distribution as if we sampled a Mondrian process directly on that subdomain.

More precisely, let $m \sim \mathrm{MP}(\lambda, \Theta_1, \ldots, \Theta_D)$ and, for each dimension $d$, let $\Phi_d$ be a connected subdomain of $\Theta_d$. The *restriction* $\rho(m, \Phi_1, \ldots, \Phi_D)$ of $m$ to $\Phi_1 \times \cdots \times \Phi_D$ is the subtree of cuts within $\Phi_1 \times \cdots \times \Phi_D$. We define restrictions inductively: If there are no cuts in $m$, i.e. $m = \Theta_1 \times \cdots \times \Theta_D$, then $\rho(m, \Phi_1, \ldots, \Phi_D)$ is simply $\Phi_1 \times \cdots \times \Phi_D$. Otherwise $m = \langle d, x, \lambda, m_<, m_> \rangle$ for some $d$, $x$, and $\lambda$, and where $m_<$ and $m_>$ are the two subtrees. Let $\Theta_d^{<x}$, $\Theta_d^{>x}$ be the $d$'th domains of $m_<$ and $m_>$ respectively. If $x \notin \Phi_d$ this implies that $\Phi_d$ must be on exactly one side of $x$ (because $\Phi_d$ and $\Theta_d$ are connected). W.l.o.g., assume $\Phi_d \subset \Theta_d^{<x}$. In this case, $\rho(m, \Phi_1, \ldots, \Phi_D) = \rho(m_<, \Phi_1, \ldots, \Phi_D)$. If $x \in \Phi_d$ then both $\Theta_d^{<x}$ and $\Theta_d^{>x}$ overlap $\Phi_d$ and $\rho(m, \Phi_1, \ldots, \Phi_D) = \langle d, x, \lambda, \rho(m_<, \Phi_1, \ldots, \Phi_d \cap \Theta_d^{<x}, \ldots \Phi_D), \rho(m_>, \Phi_1, \ldots, \Phi_d \cap \Theta_d^{>x}, \ldots \Phi_D) \rangle$.

By integrating out the variables on nodes not contained in the restriction, it can be shown that the restriction $\rho(m, \Phi_1, \ldots, \Phi_D)$ is itself distributed according to a Mondrian $\mathrm{MP}(\lambda, \Phi_1, \ldots, \Phi_D)$.

So far the construction of the Mondrian process assumes that each domain $\Theta_d$ has finite measure. A consequence of this consistency property is that we can now use the Daniell-Kolmogorov extension theorem to extend the Mondrian process to $\sigma$-finite domains (those that can be written as a countable union of finite domains). For example, from a Mondrian process on products of intervals, we can construct a Mondrian process on all of $\mathbb{R}^D$. Note that if the domains have infinite measure, the tree of cuts will be infinitely deep with no root and infinitely many leaves (being the infinite partition of the product space). However the restriction of the tree to any given finite subdomains will be finite with a root (with probability one).

**Mondrian Slices**: One interesting specific case of consistency under restriction is worth mentioning. Suppose that our subdomains are $\Phi_1 = \{y\}$ and $\Phi_d = \Theta_d$ for $d \geq 2$. That is, we consider the restriction of the Mondrian to a slice of the space where the first coordinate takes on value $y$. The consistency property shows that the restriction $\rho = \rho(m, \Phi_1, \ldots, \Phi_D)$ onto these subdomains is distributed according to a Mondrian as well. But since $\mu_1$ is non-atomic, $\mu_1(\{y\}) = 0$ thus $\rho$ will not have any cuts in the first domain (with probability 1). That is, we can interpret $\rho$ as a draw from a $D - 1$ dimensional Mondrian with domains $\Theta_2, \ldots, \Theta_D$. This is true of any lower dimensional slice of the Mondrian. One particular extreme is that since a one dimensional Mondrian is simply the

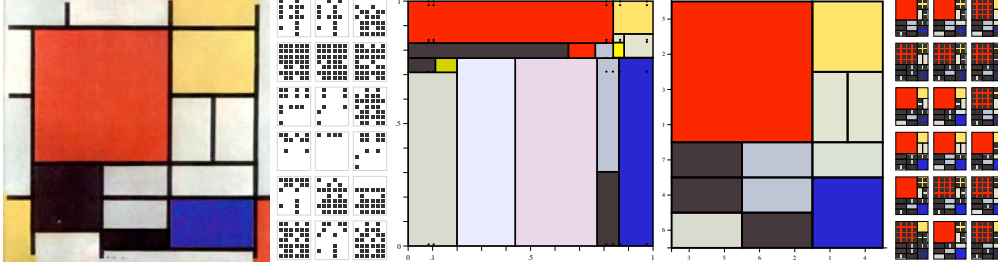

**Figure 2:** Modeling a Mondrian with a Mondrian: A posterior sample given relational data created from an actual Mondrian painting. (from left) (1) Composition with Large Blue Plane, Red, Black, Yellow, and Gray (1921). (2) Raw relational data, randomly shuffled. These synthetic data were generated by fitting a regular $6 \times 7$ point array over the painting (6 row objects, 7 column objects), and using the blocks in the painting to determine the block structure of these 42 relations. We then sampled 18 relational arrays with this block structure. (3) Posterior sample of Mondrian process on unit square. The colors are for visual effect only as the partitions are contiguous rectangles. The small black dots are the embedding of the pairs $(\xi_i, \eta_j)$ into the unit square. Each point represents a relation $R_{i,j}$; each row of points are the relations $(R_{i,\cdot})$ for an object $\xi_i$, and similarly for columns. Relations in the same block are clustered together. (4) Induced partition on the (discrete) relational array, matching the painting. (5) Partitioned and permuted relational data showing block structure.

break-and-branch generative process for a Poisson process, any one dimensional slice of a Mondrian gives a Poisson point process.

**Conditional Mondrians**: Using the consistency property, we can derive the conditional distribution of a Mondrian $m$ with rate $\lambda$ on $\Theta_1 \times \cdots \times \Theta_D$ given its restriction $\rho = \rho(m, \Phi_1, \ldots, \Phi_D)$. To do so, we have to consider three possibilities: when $m$ contains no cuts, when the first cut of $m$ is in $\rho$, and when the first cut of $m$ is above $\rho$. Fortunately the probabilities of each of these events can be computed easily, and amounts to drawing an exponential sample $E \sim \mathrm{Exp}(\sum_d \mu_d(\Theta_d \setminus \Phi_d))$, and comparing it against the diminished rate after the first cut in $\rho$. Pseudocode for generating from a conditional Mondrian is given in Algorithm 1. When every domain of $\rho$ has zero measure, i.e., $\mu_d(\Phi_d) = 0$ for all $d$, the conditional Mondrian reduces to an unconditional Mondrian.

---

**Algorithm 1** Conditional Mondrian $m \sim \mathrm{MP}(\lambda, \Theta_1, \ldots, \Theta_D \mid \rho)$ $\qquad \rho = \phi_d = \emptyset$ is unconditioned

---

1. **let** $\lambda' \leftarrow \lambda - E$ where $E \sim \mathrm{Exp}(\sum_{d=1}^{D} \mu_d(\Theta_d \setminus \Phi_d))$.
2. **if** $\rho$ has no cuts **then** $\lambda'' \leftarrow 0$ **else** $\langle d', x', \lambda'', \rho_<, \rho_> \rangle \leftarrow \rho$.
3. **if** $\lambda' < \lambda''$ **then** *take root form of* $\rho$
4.     **if** $\rho$ has no cut **then**
5.         **return** $m \leftarrow \Theta_1 \times \cdots \times \Theta_D$.
6.     **else**   $(d', x')$ *is the first cut in* $m$
7.         **return** $m \leftarrow \langle d', x', \lambda'', \mathrm{MP}(\lambda'', \Theta_1, \ldots, \Theta_{d'}^{<x'}, \ldots, \Theta_D \mid \rho_<),$
                                   $\mathrm{MP}(\lambda'', \Theta_1, \ldots, \Theta_{d'}^{>x'}, \ldots, \Theta_D \mid \rho_>) \rangle$.
8. **else**   $\lambda'' < \lambda'$ *and there is a cut in* $m$ *above* $\rho$
9.     draw a cut $(d, x)$ outside $\rho$, i.e., $p(d) \propto \mu_d(\Theta_d \setminus \Phi_d)$, $x \mid d \sim \frac{\mu_d}{\mu_d(\Theta_d \setminus \Phi_d)}$
        *without loss of generality suppose* $\Phi_d \subset \Theta_d^{<x}$
10.   **return** $m \leftarrow \langle d, x, \lambda', \mathrm{MP}(\lambda', \Theta_1, \ldots, \Theta_d^{<x}, \ldots, \Theta_D \mid \rho),$
                             $\mathrm{MP}(\lambda', \Theta_1, \ldots, \Theta_d^{>x}, \ldots, \Theta_D) \rangle$.

---

**Partition Structure**: The Mondrian is simple enough that we can characterize a number of its other properties. As an example, the expected number of slices along each dimension of $(0, A) \times (0, B)$ is $\lambda A$ and $\lambda B$, while the expected total number of partitions is $(1 + \lambda A)(1 + \lambda B)$. Interestingly, this is also the expected number of partitions in a biclustering model where we first have two independent Poisson processes with rate $\lambda$ partition $(0, A)$ and $(0, B)$, and then form the product partition of $(0, A) \times (0, B)$.

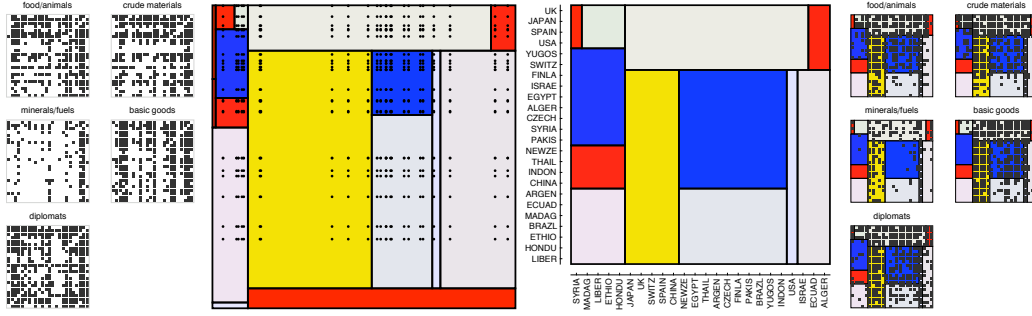

**Figure 3:** Trade and Diplomacy relations between 24 countries in 1984. $R_{ij} = 1$ (black squares) implies that country $i$ imports $R$ from country $j$. The colors are for visual effect only as the partitions are contiguous rectangles.

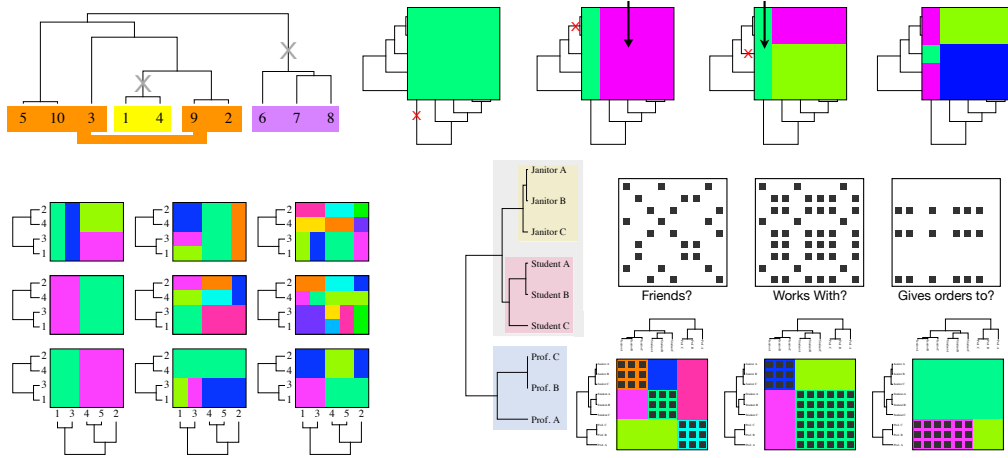

**Figure 4:** (clockwise from bottom left) (1) Nine samples from the Mondrian process on Kingman coalescents with rate $\lambda = 0.25, 0.5$, and 1, respectively. As the rate increases, partitions become finer. Note that partitions are not necessarily contiguous; we use color to identify partitions. The partition structure is related to the annotated hierarchies model (Roy et al., 2007). (2) Kingman (1982a,b) describes the relationship between random trees and the DP, which we exploit to define a nonparametric, hierarchical block model. (3) A sequence of cuts; each cut separates a subtree. (4) Posterior trees and Mondrian processes on a synthetic social network.

## 5 Relational Modeling

To illustrate how the Mondrian process can be used to model relational data, we describe two non-parametric block models for exchangeable relations. While we will only consider binary data and assume that each block is conditionally iid, the ideas can be extended to many likelihood models.

Recall the Aldous-Hoover representation $(\theta, \xi_i, \eta_j, p_R)$ for exchangeable arrays. Using a Mondrian process with *beta Lévy* measure $\mu(dx) = \alpha x^{-1} dx$, we first sample a random partition of the unit square into blocks and assign each block a probability:

$$M \sim \mathrm{MP}(\lambda, [0,1], [0,1]) \qquad \textit{slices up unit square into blocks} \qquad (6)$$

$$\phi_S \,|\, M \sim \mathrm{Beta}(a_0, a_1), \,\forall S \in M. \qquad \textit{each block } S \textit{ gets a probability } \phi_S \qquad (7)$$

The pair $(M, \phi)$ plays the role of $\theta$ in the Aldous-Hoover representation. We next sample row and column representations ($\xi_i$ and $\eta_j$, respectively), which have a geometrical interpretation as $x,y$-coordinates $(\xi_i, \eta_j)$ in the unit square:

$$\xi_i \sim \mathrm{U}[0,1], \, i \in \{1, \ldots, n\} \qquad \textit{shared x coordinate for each row} \qquad (8)$$

$$\eta_j \sim \mathrm{U}[0,1], \, j \in \{1, \ldots, n\}. \qquad \textit{shared y coordinate for each column} \qquad (9)$$

Let $S_{ij}$ be the block $S \in M$ such that $(\xi_i, \eta_j) \in S$. We finally sample the array $R$ of relations:

$$R_{ij} \,|\, \xi, \eta, \phi, M \sim \mathrm{Bernoulli}(\phi_{S_{ij}}), i, j \in \{1, \ldots, n\}. \qquad R_{ij} \textit{ is true w.p. } \phi_{S_{ij}} \qquad (10)$$

This model clusters relations together whose $(\xi_i, \eta_j)$ pairs fall in the same blocks in the Mondrian partition and models each cluster with a beta-binomial likelihood model. By mirroring the Aldous-Hoover representation, we guarantee that $R$ is exchangeable and that there is no order dependence.

This model is closely related to the IRM (Kemp et al., 2006) and IHRM (Xu et al., 2006), where rows and columns are first clustered using a CRP prior, then each relation $R_{ij}$ is conditionally independent from others given the clusters that row $i$ and column $j$ belong to. In particular, if we replace Eq. (6) with

$$M \sim \text{MP}(\lambda, [0, 1]) \times \text{MP}(\lambda, [0, 1]), \qquad \textit{product of partitions of unit intervals} \qquad (11)$$

then we recover the same marginal distribution over relations as the IRM/IHRM. To see this, recall that a Mondrian process in one-dimension produces a partition whose cut points follow a Poisson point process. Teh et al. (2007) show that the stick lengths (i.e., partitions) induced by a Poisson point process on $[0, 1]$ with the beta Lévy measure have the same distribution as those in the stick-breaking construction of the DP. Therefore, (11) is the product of two stick-breaking priors. In comparison, any one dimensional slice of (6), e.g., each column or row of the relation, is marginally distributed as a DP, but is more flexible than the product of one-dimensional Mondrian processes.

We can also construct an exchangeable variant of the Annotated Hierarchies model (a hierarchical block model) by moving from the unit square to a product of random trees drawn from Kingman's coalescent prior (Kingman, 1982a). Let $\mu_d$ be Lebesgue measure.

$$T_d \sim \text{KC}(\lambda), \forall d \in \{1, \ldots, D\} \qquad \textit{for each dimension, sample a tree} \qquad (12)$$
$$M \,|\, T \sim \text{MP}(2\alpha, T_1, \ldots, T_D) \qquad \textit{partition the cross product of trees} \qquad (13)$$
$$\phi_S \,|\, M \sim \text{Beta}(a_0, a_1), \forall S \in M. \qquad \textit{each block } S \textit{ gets a probability } \phi_S \qquad (14)$$

Let $S_{ij}$ be the subset $S \in M$ where leaves $(i, j)$ fall in $S$. Then

$$R_{ij} \,|\, \phi, M \sim \text{Bernoulli}(\phi_{S_{ij}}), i, j \in \{1, \ldots, n\}. \qquad R_{ij} \textit{ is true w.p. } \phi_{S_{ij}} \qquad (15)$$

Figure 4 shows some samples from this prior. Again, this model is related to the DP. Kingman shows that the partition on the leaves of a coalescent tree when its edges are cut by a Poisson point process is the same as that of a DP (Figure 4). Therefore, the partition structure along every row and column is marginally the same as a DP. Both the unit square and product of random trees models give DP distributed partitions on each row and column, but they have different inductive biases.

## 6 Experiments

The first data set was synthetically created using an actual painting by Piet Mondrian, whose grid-based paintings were the inspiration for the name of this process. Using the model defined by (10) and a uniform rate measure, we performed a Markov chain Monte Carlo (MCMC) simulation of the posterior distribution over the Mondrian, $\xi$'s, $\eta$'s, and hyperparameters. We employed a number of Metropolis-Hastings (MH) proposals that rotated, scaled, flipped, and resampled portions of the Mondrian. It can be shown that the conditional distribution of each $\xi_i$ and $\eta_j$ is piecewise constant; given the conjugacy of the beta-binomial, we can Gibbs sample the $\xi$'s and $\eta$'s. Figure 2 shows a sample after 1500 iterations (starting from a random initialization) where the partition on the array is exactly recovered. This was a typical attractor state for random initializations. While the data are sufficient to recover the partition on the array, they are not sufficient to recover the underlying Mondrian process. It is an open question as to its identifiability in the limit of infinite data.

We next analyzed the classic *Countries* data set from the network analysis literature (Wasserman and Faust, 1994), which reports trade in 1984 between 24 countries in food and live animals; crude materials; minerals and fuels; basic manufactured goods; and exchange of diplomats. We applied the model defined by (10). Figure 3 illustrates the type of structure the model uncovers during MCMC simulation; it has recognized several salient groups of countries acting in blocs; e.g., Japan, the UK, Switzerland, Spain and China export to nearly all countries, although China behaves more like the other Pacific Rim countries as an importer. The diplomats relation is nearly symmetric, but the model does not represent symmetry explicitly and must redundantly learn the entire relation. Reflecting the Mondrian about the line $y = x$ is one way to enforce symmetry in the partition.

In our final experiment, we analyzed a synthetic social network consisting of nine university employees: 3 janitors, 3 professors and 3 students. Given three relations (friends, works-with, and

gives-orders-to), the maximum a posteriori Mondrian process partitions the relations into homogeneous blocks. Tree structures around the MAP clustered the janitors, professors and students into three close-knit groups, and preferred to put the janitors and students more closely together in the tree. Inference in this model is particularly challenging given the large space of trees and partitions.

## 7 Discussion

While the Mondrian process has many elegant properties, much more work is required to determine its usefulness for relational modeling. Just as effective inference procedures preceded the popularity of the Dirichlet process, a similar leap in inference sophistication will be necessary to assess the Mondrian process on large data sets. We are currently investigating improved MCMC sampling schemes for the Mondrian process, as well as working to develop a combinatorial representation of the distribution on partitions induced by the Mondrian process. Such a representation is of practical interest (possibly leading to improved inference schemes) and of theoretical interest, being a multidimensional generalization of Chinese restaurant processes.

The axis-aligned partitions of $[0, 1]^n$ produced by the Mondrian process have been studied extensively in combinatorics and computational geometry, where they are known as *guillotine partitions*. Guillotine partitions have wide ranging applications including circuit design, approximation algorithms and computer graphics. However, the question of consistent stochastic processes over guillotine partitions, i.e. the question addressed here, has not, to our knowledge, been studied before.

At a high level, we believe that developing nonparametric priors on complex data structures from computer science may successfully bridge the gap between old-fashioned Artificial Intelligence and modern statistical approaches. Developing representations for these typically recursive structures will require us to go beyond graphical models; stochastic lambda calculus is an appealing option.

## Footnotes

[1] We consider binary relations here but the ideas generalize easily to multidimensional relations.

[2] Holland et al. (1983) introduced stochastic block models. Recent variations (Kemp et al., 2006; Xu et al., 2006; Roy et al., 2007) descend from Wasserman and Anderson (1987) and Nowicki and Snijders (2001).

[3]In this paper we shall always mean infinite exchangeability when we state exchangeability.

## References

D. J. Aldous. Representations for Partially Exchangeable Arrays of Random Variables. *Journal of Multivariate Analysis*, 11:581–598, 1981.

J. M. Bernardo and A. F. M. Smith. *Bayesian theory*. John Wiley & Sons, 1994.

B. de Finetti. Funzione caratteristica di un fenomeno aleatorio. *Atti della R. Academia Nazionale dei Lincei, Serie 6. Memorie, Classe di Scienze Fisiche, Mathematice e Naturale*, 4:251299, 1931.

P. Diaconis and S. Janson. Graph limits and exchangeable random graphs. arXiv:0712.2749v1, 2007.

P. W. Holland, K. B. Laskey, and S. Leinhardt. Stochastic blockmodels: First steps. *Social Networks*, 5(2):109 − 137, 1983.

D. Hoover. Relations on probability spaces and arrays of random variables. Technical report, Preprint, Institute for Advanced Study, Princeton, NJ, 1979.

O. Kallenberg. *Probabilistic Symmetries and Invariance Principles*. Springer, 2005.

C. Kemp, J. Tenenbaum, T. Griffiths, T. Yamada, and N. Ueda. Learning systems of concepts with an infinite relational model. In *Proceedings of the 21st National Conference on Artificial Intelligence*, 2006.

J. F. C. Kingman. On the genealogy of large populations. *Journal of Applied Probability*, 19:27–43, 1982a.

J. F. C. Kingman. The coalescent. *Stochastic Processes and their Applications*, 13:235–248, 1982b.

L. Lovász and B. Szegedy. Limits of dense graph sequences. *J. Comb. Theory B*, 96:933957, 2006.

K. Nowicki and T. A. B. Snijders. Estimation and prediction for stochastic blockstructures. *Journal of the American Statistical Association*, 96:1077–1087(11), 2001.

D. M. Roy, C. Kemp, V. Mansinghka, and J. B. Tenenbaum. Learning annotated hierarchies from relational data. In *Advances in Neural Information Processing Systems 19*, 2007.

C. Ryll-Nardzewski. On stationary sequences of random variables and the de Finetti's equivalence. *Colloq. Math.*, 4:149–156, 1957.

J. Sethuraman. A Constructive definition of Dirichlet priors. *Statistica Sinica*, 4:639–650, 1994.

Y. W. Teh, D. Görür, and Z. Ghahramani. Stick-breaking construction for the Indian buffet process. In *Proceedings of the International Conference on Artificial Intelligence and Statistics*, volume 11, 2007.

S. Wasserman and C. Anderson. Stochastic a posteriori blockmodels: Construction and assessment. *Social Networks*, 9(1):1 − 36, 1987.

S. Wasserman and K. Faust. *Social Network Analysis: Methods and Applications*, pages 64–65. Cambridge University Press, 1994.

Z. Xu, V. Tresp, K. Yu, and H.-P. Kriegel. Infinite Hidden Relational Models. In *Proceedings of the 22nd Conference on Uncertainty in Artificial Intelligence*, 2006.

